# Nonparametric Latent Feature Models for Link Prediction

**Kurt T. Miller**
EECS
University of California
Berkeley, CA 94720
tadayuki@cs.berkeley.edu

**Thomas L. Griffiths**
Psychology and Cognitive Science
University of California
Berkeley, CA 94720
tom_griffiths@berkeley.edu

**Michael I. Jordan**
EECS and Statistics
University of California
Berkeley, CA 94720
jordan@cs.berkeley.edu

## Abstract

As the availability and importance of relational data—such as the friendships summarized on a social networking website—increases, it becomes increasingly important to have good models for such data. The kinds of latent structure that have been considered for use in predicting links in such networks have been relatively limited. In particular, the machine learning community has focused on latent class models, adapting Bayesian nonparametric methods to jointly infer how many latent classes there are while learning which entities belong to each class. We pursue a similar approach with a richer kind of latent variable—latent features—using a Bayesian nonparametric approach to simultaneously infer the number of features at the same time we learn which entities have each feature. Our model combines these inferred features with known covariates in order to perform link prediction. We demonstrate that the greater expressiveness of this approach allows us to improve performance on three datasets.

## 1 Introduction

Statistical analysis of social networks and other relational data has been an active area of research for over seventy years and is becoming an increasingly important problem as the scope and availability of social network datasets increase [1]. In these problems, we observe the interactions between a set of entities and we wish to extract informative representations that are useful for making predictions about the entities and their relationships. One basic challenge is link prediction, where we observe the relationships (or "links") between some pairs of entities in a network (or "graph") and we try to predict unobserved links. For example, in a social network, we might only know some subset of people are friends and some are not, and seek to predict which other people are likely to get along.

Our goal is to improve the expressiveness and performance of generative models based on extracting latent structure representing the properties of individual entities from the observed data, so we will focus on these kinds of models. This rules out approaches like the popular $p^*$ model that uses global quantities of the graph, such as how many edges or triangles are present [2, 3]. Of the approaches that do link prediction based on attributes of the individual entities, these can largely be classified into class-based and feature-based approaches. There are many models that can be placed under these approaches, so we will focus on the models that are most comparable to our approach.

Most generative models using a class-based representation are based on the stochastic blockmodel, introduced in [4] and further developed in [5]. In the most basic form of the model, we assume there are a finite number of classes that entities can belong to and that these classes entirely determine the structure of the graph, with the probability of a link existing between two entities depending only on the classes of those entities. In general, these classes are unobserved, and inference reduces to assigning entities to classes and inferring the class interactions. One of the important issues that arise in working with this model is determining how many latent classes there are for a given problem. The Infinite Relational Model (IRM) [6] used methods from nonparametric Bayesian statistics to tackle this problem, allowing the number of classes to be determined at inference time. The Infinite Hidden Relational Model [7] further elaborated on this model and the Mixed Membership Stochastic Blockmodel (MMSB) [8] extended it to allow entities to have mixed memberships.

All these class-based models share a basic limitation in the kinds of relational structure they naturally capture. For example, in a social network, we might find a class which contains "male high school athletes" and another which contains "male high school musicians." We might believe these two classes will behave similarly, but with a class-based model, our options are to either merge the classes or duplicate our knowledge about common aspects of them. In a similar vein, with a limited amount of data, it might be reasonable to combine these into a single class "male high school students," but with more data we would want to split this group into athletes and musicians. For every new attribute like this that we add, the number of classes would potentially double, quickly leading to an overabundance of classes. In addition, if someone is both an athlete and a musician, we would either have to add another class for that or use a mixed membership model, which would say that the more a student is an athlete, the less he is a musician.

An alternative approach that addresses this problem is to use features to describe the entities. There could be a separate feature for "high school student," "male," "athlete," and "musician" and the presence or absence of each of these features is what defines each person and determines their relationships. One class of latent-feature models for social networks has been developed by [9, 10, 11], who proposed real-valued vectors as latent representations of the entities in the network where depending on the model, either the distance, inner product, or weighted combination of the vectors corresponding to two entities affects the likelihood of there being a link between them. However, extending our high school student example, we might hope that instead of having arbitrary real-valued features (which are still useful for visualization), we would infer binary features where each feature could correspond to an attribute like "male" or "athlete." Continuing our earlier example, if we had a limited amount of data, we might not pick up on a feature like "athlete." However, as we observe more interactions, this could emerge as a clear feature. Instead of doubling the numbers of classes in our model, we simply add an additional feature. Determining the number of features will therefore be of extreme importance.

In this paper, we present the *nonparametric latent feature relational model*, a Bayesian nonparametric model in which each entity has binary-valued latent features that influences its relations. In addition, the relations depend on a set of known covariates. This model allows us to simultaneously infer how many latent features there are while at the same time inferring what features each entity has and how those features influence the observations. This model is strictly more expressive than the stochastic blockmodel. In Section 2, we describe a simplified version of our model and then the full model. In Section 3, we discuss how to perform inference. In Section 4, we illustrate the properties of our model using synthetic data and then show that the greater expressiveness of the latent feature representation results in improved link prediction on three real datasets. Finally, we conclude in Section 5.

## 2 The nonparametric latent feature relational model

Assume we observe the directed relational links between a set of $N$ entities. Let $Y$ be the $N \times N$ binary matrix that contains these links. That is, let $y_{ij} \equiv Y(i, j) = 1$ if we observe a link from entity $i$ to entity $j$ in that relation and $y_{ij} = 0$ if we observe that there is not a link. Unobserved links are left unfilled. Our goal will be to learn a model from the observed links such that we can predict the values of the unfilled entries.

## 2.1 Basic model

In our basic model, each entity is described by a set of binary features. We are not given these features a priori and will attempt to infer them. We assume that the probability of having a link from one entity to another is entirely determined by the combined effect of all pairwise feature interactions. If there are $K$ features, then let $Z$ be the $N \times K$ binary matrix where each row corresponds to an entity and each column corresponds to a feature such that $z_{ik} \equiv Z(i, k) = 1$ if the $i^{\text{th}}$ entity has feature $k$ and $z_{ik} = 0$ otherwise. and let $Z_i$ denote the feature vector corresponding to entity $i$. Let $W$ be a $K \times K$ real-valued weight matrix where $w_{kk'} \equiv W(k, k')$ is the weight that affects the probability of there being a link from entity $i$ to entity $j$ if both entity $i$ has feature $k$ and entity $j$ has feature $k'$.

We assume that links are independent conditioned on $Z$ and $W$, and that only the features of entities $i$ and $j$ influence the probability of a link between those entities. This defines the likelihood

$$\Pr(Y|Z, W) \quad = \quad \prod_{i,j} \Pr(y_{ij}|Z_i, Z_j, W) \tag{1}$$

where the product ranges over all pairs of entities. Given the feature matrix $Z$ and weight matrix $W$, the probability that there is a link from entity $i$ to entity $j$ is

$$\Pr(y_{ij} = 1|Z, W) = \sigma\left(Z_i W Z_j^\top\right) = \sigma\left(\sum_{k,k'} z_{ik} z_{jk'} w_{kk'}\right) \tag{2}$$

where $\sigma(\cdot)$ is a function that transforms values on $(-\infty, \infty)$ to $(0, 1)$ such as the sigmoid function $\sigma(x) = \frac{1}{1+\exp(-x)}$ or the probit function $\sigma(x) = \Phi(x)$. An important aspect of this model is that all-zero columns of $Z$ do not affect the likelihood. We will take advantage of this in Section 2.2.

This model is very flexible. With a single feature per entity, it is equivalent to a stochastic block-model. However, since entities can have more than a single feature, the model is more expressive. In the high school student example, each feature can correspond to an attribute like "male," "musician," and "athlete." If we were looking at the relation "friend of" (not necessarily symmetric!), then the weight at the (athlete, musician) entry of $W$ would correspond to the weight that an athlete would be a friend of a musician. A positive weight would correspond to an increased probability, a negative weight a decreased probability, and a zero weight would indicate that there is no correlation between those two features and the observed relation. The more positively correlated features people have, the more likely they are to be friends. Another advantage of this representation is that if our data contained observations of students in two distant locations, we could have a geographic feature for the different locations. While other features such as "athlete" or "musician" might indicate that one person could be a friend of another, the geographic features could have extremely negative weights so that people who live far from each other are less likely to be friends. However, the parameters for the non-geographic features would still be tied for all people, allowing us to make stronger inferences about how they influence the relations. Class-based models would need an abundance of classes to capture these effects and would not have the same kind of parameter sharing.

Given the full set of observations $Y$, we wish to infer the posterior distribution of the feature matrix $Z$ and the weights $W$. We do this using Bayes' theorem, $p(Z, W|Y) \propto p(Y|Z, W)p(Z)p(W)$, where we have placed an independent prior on $Z$ and $W$. Without any prior knowledge about the features or their weights, a natural prior for $W$ involves placing an independent $N(0, \sigma_w^2)$ prior on each $w_{ij}$. However, placing a prior on $Z$ is more challenging. If we knew how many features there were, we could place an arbitrary parametric prior on $Z$. However, we wish to have a flexible prior that allows us to simultaneously infer the number of features at the same time we infer all the entries in $Z$. The Indian Buffet Process is such a prior.

## 2.2 The Indian Buffet Process and the basic generative model

As mentioned in the previous section, any features which are all-zero do not affect the likelihood. That means that even if we added an infinite number of all-zero features, the likelihood would remain the same. The Indian Buffet Process (IBP) [12] is a prior on infinite binary matrices such that with probability one, a feature matrix drawn from it for a finite number of entities will only have a finite number of non-zero features. Moreover, any feature matrix, no matter how many non-zero features

it contains, has positive probability under the IBP prior. It is therefore a useful nonparametric prior to place on our latent feature matrix $Z$.

The generative process to sample matrices from the IBP can be described through a culinary metaphor that gave the IBP its name. In this metaphor, each row of $Z$ corresponds to a diner at an Indian buffet and each column corresponds to a dish at the infinitely long buffet. If a customer takes a particular dish, then the entry that corresponds to the customer's row and the dish's column is a one and the entry is zero otherwise. The culinary metaphor describes how people choose the dishes. In the IBP, the first customer chooses a Poisson($\alpha$) number of dishes to sample, where $\alpha$ is a parameter of the IBP. The $i^{\text{th}}$ customer tries each previously sampled dish with probability proportional to the number of people that have already tried the dish and then samples a Poisson($\alpha/i$) number of new dishes. This process is exchangeable, which means that the order in which the customers enter the restaurant does not affect the configuration of the dishes that people try (up to permutations of the dishes as described in [12]). This insight leads to a straightforward Gibbs sampler to do posterior inference that we describe in Section 3.

Using an IBP prior on $Z$, our basic generative latent feature relational model is:

$$Z \sim \text{IBP}(\alpha)$$
$$w_{kk'} \sim \mathcal{N}(0, \sigma_w^2) \qquad \text{for all } k, k' \text{ for which features } k \text{ and } k' \text{ are non-zero}$$
$$y_{ij} \sim \sigma\left(Z_i W Z_j^\top\right) \qquad \text{for each observation.}$$

### 2.3 Full nonparametric latent feature relational model

We have described the basic nonparametric latent feature relational model. We now combine it with ideas from the social network community to get our full model. First, we note that there are many instances of logit models used in statistical network analysis that make use of covariates in link prediction [2]. Here we will focus on a subset of ideas discussed in [10]. Let $X_{ij}$ be a vector that influences the relation $y_{ij}$, let $X_{p,i}$ be a vector of known attributes of entity $i$ when it is the parent of a link, and let $X_{c,i}$ be a vector of known attributes of entity $i$ when it is a child of a link. For example, in Section 4.2, when $Y$ represents relationships amongst countries, $X_{ij}$ is a scalar representing the geographic similarity between countries ($X_{ij} = \exp(-d(i,j))$) since this could influence the relationships and $X_{p,i} = X_{c,i}$ is a set of known features associated with each country ($X_{p,i}$ and $X_{c,i}$ would be distinct if we had covariates specific to each country's roles). We then let $c$ be a normally distributed scalar and $\beta$, $\beta_p$, $\beta_c$, $a$, and $b$ be normally distributed vectors in our full model in which

$$\Pr(y_{ij} = 1 | Z, W, X, \beta, a, b, c) = \sigma\left(Z_i W Z_j^\top + \beta^\top X_{ij} + (\beta_p^\top X_{p,i} + a_i) + (\beta_c^\top X_{c,j} + b_j) + c\right). \quad (3)$$

If we do not have information about one or all of $X$, $X_p$, and $X_c$, we drop the corresponding term(s). In this model, $c$ is a global offset that affects the default likelihood of a relation and $a_i$ and $b_j$ are entity and role specific offsets.

So far, we have only considered the case of observing a single relation. It is not uncommon to observe multiple relations for the same set of entities. For example, in addition to the "friend of" relation, we might also observe the "admires" and "collaborates with" relations. We still believe that each entity has a single set of features that determines all its relations, but these features will not affect each relation in the same way. If we are given $m$ relations, label them $Y^1, Y^2, \ldots, Y^m$. We will use the same features for each relation, but we will use an independent weight matrix $W^i$ for each relation $Y^i$. In addition, covariates might be relation specific or common across all relations. Regardless, they will interact in different ways in each relation. Our full model is now

$$\Pr(Y^1, \ldots, Y^m | Z, \{W^i, X^i, \beta^i, a^i, b^i, c^i\}_{i=1}^m) = \prod_{i=1}^m \Pr(Y^i | Z, W^i, X^i, \beta^i, a^i, b^i, c^i).$$

### 2.4 Variations of the nonparametric latent feature relational model

The model that we have defined is for directed graphs in which the matrix $Y^i$ is not assumed to be symmetric. For undirected graphs, we would like to define a symmetric model. This is easy to do by restricting $W^i$ to be symmetric. If we further believe that the features we learn should not interact, we can assume that $W^i$ is diagonal.

## 2.5 Related nonparametric latent feature models

There are two models related to our nonparametric latent feature relational model that both use the IBP as a prior on binary latent feature matrices. The most closely related model is the Binary Matrix Factorization (BMF) model of [13]. The BMF is a general model with several concrete variants, the most relevant of which was used to predict unobserved entries of binary matrices for image reconstruction and collaborative filtering. If $Y$ is the observed part of a binary matrix, then in this variant, we assume that $Y|U, V, W \sim \sigma(UWV^\top)$ where $\sigma(\cdot)$ is the logistic function, $U$ and $V$ are independent binary matrices drawn from the IBP, and the entries in $W$ are independent draws from a normal distribution. If $Y$ is an $N \times N$ matrix where we assume the rows and columns have the same features (i.e., $U = V$), then this special case of their model is equivalent to our basic (covariate-free) model. While [13] were interested in a more general formalization that is applicable to other tasks, we have specialized and extended this model for the task of link prediction. The other related model is the ADCLUS model [14]. This model assumes we are given a symmetric matrix of nonnegative similarities $Y$ and that $Y = ZWZ^\top + \epsilon$ where $Z$ is drawn from the IBP, $W$ is a diagonal matrix with entries independently drawn from a Gamma distribution, and $\epsilon$ is independent Gaussian noise. This model does not allow for arbitrary feature interactions nor does it allow for negative feature correlations.

## 3 Inference

Exact inference in our nonparametric latent feature relational model is intractable [12]. However, the IBP prior lends itself nicely to approximate inference via Markov Chain Monte Carlo [15]. We first describe inference in the single relation, basic model, later extending it to the full model. In our basic model, we must do posterior inference on $Z$ and $W$. Since with probability one, any sample of $Z$ will have a finite number of non-zero entries, we can store just the non-zero columns of each sample of the infinite binary matrix $Z$. Since we do not have a conjugate prior on $W$, we must also sample the corresponding entries of $W$. Our sampler is as follows:

**Given $W$, resample $Z$**   We do this by resampling each row $Z_i$ in succession. When sampling entries in the $i^{\text{th}}$ row, we use the fact that the IBP is exchangeable to assume that the $i^{\text{th}}$ customer in the IBP was the last one to enter the buffet. Therefore, when resampling $z_{ik}$ for non-zero columns $k$, if $m_k$ is the number of non-zero entries in column $k$ excluding row $i$, then

$$\Pr(z_{ik} = 1 | Z_{-ik}, W, Y) \quad \propto \quad m_k \Pr(Y | z_{ik} = 1, Z_{-ik}, W).$$

We must also sample $z_{ik}$ for each of the infinitely many all-zero columns to add features to the representation. Here, we use the fact that in the IBP, the prior distribution on the number of new features for the last customer is Poisson($\alpha/N$). As described in [12], we must then weight this by the likelihood term for having that many new features, computing this for $0, 1, \ldots . k_{\max}$ new features for some maximum number of new features $k_{\max}$ and sampling the number of new features from this normalized distribution. The main difficulty arises because we have not sampled the values of $W$ for the all-zero columns and we do not have a conjugate prior on $W$, so we cannot compute the likelihood term exactly. We can adopt one of the non-conjugate sampling approaches from the Dirichlet process [16] to this task or use the suggestion in [13] to include a Metropolis-Hastings step to propose and either accept or reject some number of new columns and the corresponding weights. We chose to use a stochastic Monte Carlo approximation of the likelihood. Once the number of new features is sampled, we must sample the new values in $W$ as described below.

**Given $Z$, resample $W$**   We sequentially resample each of the weights in $W$ that correspond to non-zero features and drop all weights that correspond to all-zero features. Since we do not have a conjugate prior on $W$, we cannot directly sample $W$ from its posterior. If $\sigma(\cdot)$ is the probit, we adapt the auxiliary sampling trick from [17] to have a Gibbs sampler for the entries of $W$. If $\sigma(\cdot)$ is the logistic function, no such trick exists and we resort to using a Metropolis-Hastings step for each weight in which we propose a new weight from a normal distribution centered around the old one.

**Hyperparameters**   We can also place conjugate priors on the hyperparameters $\alpha$ and $\sigma_w$ and perform posterior inference on them. We use the approach from [18] for sampling of $\alpha$.

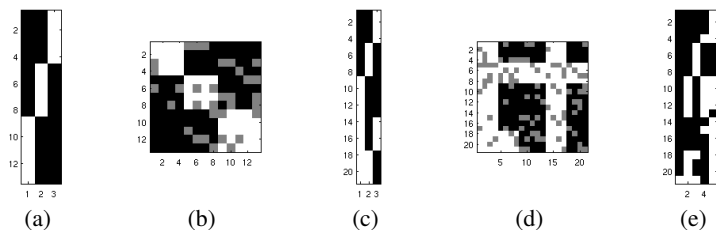

(a)        (b)        (c)        (d)        (e)

Figure 1: Features and corresponding observations for synthetic data. In (a), we show features that could be explained by a latent-class model that then produces the observation matrix in (b). White indicates one values, black indicates zero values, and gray indicates held out values. In (c), we show the feature matrix of our other synthetic dataset along with the corresponding observations in (d). (e) shows the feature matrix of a randomly chosen sample from our Gibbs sampler.

**Multiple relations** In the case of multiple relations, we can sample $W_i$ given $Z$ independently for each $i$ as above. However, when we resample $Z$, we must compute

$$\Pr(z_{ik} = 1 | Z_{-ik}, \{W, Y\}_{i=1}^m) \quad \propto \quad m_k \prod_{i=1}^m \Pr(Y^i | z_{ik} = 1, Z_{-ik}, W^i).$$

**Full model** In the full model, we must also update $\{\beta^i, \beta_p^i, \beta_c^i, a^i, b^i, c^i\}_{i=1}^m$. By conditioning on these, the update equations for $Z$ and $W^i$ take the same form, but with Equation (3) used for the likelihood. When we condition on $Z$ and $W^i$, the posterior updates for $(\beta^i, \beta_p^i, \beta_c^i, a^i, b^i, c^i)$ are independent and can be derived from the updates in [10].

**Implementation details** Despite the ease of writing down the sampler, samplers for the IBP often mix slowly due to the extremely large state space full of local optima. Even if we limited $Z$ to have $K$ columns, there are $2^{NK}$ potential feature matrices. In an effort to explore the space better, we can augment the Gibbs sampler for $Z$ by introducing split-merge style moves as described in [13] as well as perform annealing or tempering to smooth out the likelihood. However, we found that the most significant improvement came from using a good initialization. A key insight that was mentioned in Section 2.1 is that the stochastic blockmodel is a special case of our model in which each entity only has a single feature. Stochastic blockmodels have been shown to perform well for statistical network analysis, so they seem like a reasonable way to initialize the feature matrix. In the results section, we compare the performance of a random initialization to one in which $Z$ is initialized with a matrix learned by the Infinite Relational Model (IRM). To get our initialization point, we ran the Gibbs sampler for the IRM for only 15 iterations and used the resulting class assignments to seed $Z$.

## 4 Results

We first qualitatively analyze the strengths and weaknesses of our model on synthetic data, establishing what we can and cannot expect from it. We then compare our model against two class-based generative models, the Infinite Relational Model (IRM) [6] and the Mixed Membership Stochastic Blockmodel (MMSB) [8], on two datasets from the original IRM paper and a NIPS coauthorship dataset, establishing that our model does better than the best of those models on those datasets.

### 4.1 Synthetic data

We first focus on the qualitative performance of our model. We applied the basic model to two very simple synthetic datasets generated from known features. These datasets were simple enough that the basic model could attain 100% accuracy on held-out data, but were different enough to address the qualitative characteristics of the latent features inferred. In one dataset, the features were the class-based features seen in Figure 1(a) and in the other, we used the features in Figure 1(c). The observations derived from these features can be seen in Figure 1(b) and Figure 1(d), respectively.

On both datasets, we initialized $Z$ and $W$ randomly. With the very simple, class-based model, 50% of the sampled feature matrices were identical to the generating feature matrix with another 25% differing by a single bit. However, on the other dataset, only 25% of the samples were at most a single bit different than the true matrix. It is not the case that the other 75% of the samples were bad samples, though. A randomly chosen sample of $Z$ is shown in Figure 1(e). Though this matrix is different from the true generating features, with the appropriate weight matrix it predicts just as well as the true feature matrix. These tests show that while our latent feature approach is able to learn features that explain the data well, due to subtle interactions between sets of features and weights, the features themselves will not in general correspond to interpretable features. However, we can expect the inferred features to do a good job explaining the data. This also indicates that there are many local optima in the feature space, further motivating the need for good initialization.

## 4.2 Multi-relational datasets

In the original IRM paper, the IRM was applied to several datasets [6]. These include a dataset containing 54 relations of 14 countries (such as "exports to" and "protests") along with 90 given features of the countries [19] and a dataset containing 26 kinship relationships of 104 people in the Alyawarra tribe in Central Australia [20]. See [6, 19, 20] for more details on the datasets.

Our goal in applying the latent feature relational model to these datasets was to demonstrate the effectiveness of our algorithm when compared to two established class-based algorithms, the IRM and the MMSB, and to demonstrate the effectiveness of our full algorithm. For the Alyawarra dataset, we had no known covariates. For the countries dataset, $X_p = X_c$ was the set of known features of the countries and $X$ was the country distance similarity matrix described in Section 2.3.

As mentioned in the synthetic data section, the inferred features do not necessarily have any interpretable meaning, so we restrict ourselves to a quantitative comparison. For each dataset, we held out 20% of the data during training and we report the AUC, the area under the ROC (Receiver Operating Characteristic) curve, for the held-out data [21]. We report results for inferring a global set of features for all relations as described in Section 2.3 which we refer to as "global" as well as results when a different set of features is independently learned for each relation and then the AUCs of all relations are averaged together, which we refer to as "single." In addition, we tried initializing our sampler for the latent feature relational model with either a random feature matrix ("LFRM rand") or class-based features from the IRM ("LFRM w/ IRM"). We ran our sampler for 1000 iterations for each configuration using a logistic squashing function (though results using the probit are similar), throwing out the first 200 samples as burn-in. Each method was given five random restarts.

Table 1: AUC on the countries and kinship datasets. Bold identifies the best performance.

|  | Countries single | Countries global | Alyawarra single | Alyawarra global |
|---|---|---|---|---|
| LFRM w/ IRM | $0.8521 \pm 0.0035$ | $\mathbf{0.8772 \pm 0.0075}$ | $0.9346 \pm 0.0013$ | $\mathbf{0.9183 \pm 0.0108}$ |
| LFRM rand | $\mathbf{0.8529 \pm 0.0037}$ | $0.7067 \pm 0.0534$ | $\mathbf{0.9443 \pm 0.0018}$ | $0.7127 \pm 0.030$ |
| IRM | $0.8423 \pm 0.0034$ | $0.8500 \pm 0.0033$ | $0.9310 \pm 0.0023$ | $0.8943 \pm 0.0300$ |
| MMSB | $0.8212 \pm 0.0032$ | $0.8643 \pm 0.0077$ | $0.9005 \pm 0.0022$ | $0.9143 \pm 0.0097$ |

Results of these tests are in Table 1. As can be seen, the LFRM with class-based initialization outperforms both the IRM and MMSB. On the individual relations ("single"), the LFRM with random initialization also does well, beating the IRM initialization on both datasets. However, the random initialization does poorly at inferring the global features due to the coupling of features and the weights for each of the relations. This highlights the importance of proper initialization. To demonstrate that the covariates are helping, but that even without them, our model does well, we ran the global LFRM with class-based initialization without covariates on the countries dataset and the AUC dropped to $0.8713 \pm 0.0105$, which is still the best performance.

On the countries data, the latent feature model inferred on average 5-7 features when seeded with the IRM and 8-9 with a random initialization. On the kinship data, it inferred 9-11 features when seeded with the IRM and 13-19 when seeded randomly.

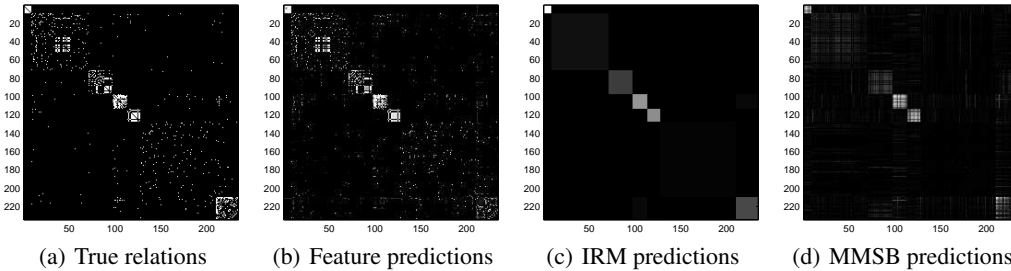

| (a) True relations | (b) Feature predictions | (c) IRM predictions | (d) MMSB predictions |

Figure 2: Predictions for all algorithms on the NIPS coauthorship dataset. In (a), a white entry means two people wrote a paper together. In (b-d), the lighter an entry, the more likely that algorithm predicted the corresponding people would interact.

## 4.3  Predicting NIPS coauthorship

As our final example, highlighting the expressiveness of the latent feature relational model, we used the coauthorship data from the NIPS dataset compiled in [22]. This dataset contains a list of all papers and authors from NIPS 1-17. We took the 234 authors who had published with the most other people and looked at their coauthorship information. The symmetric coauthor graph can be seen in Figure 2(a). We again learned models for the latent feature relational model, the IRM and the MMSB training on 80% of the data and using the remaining 20% as a test set. For the latent feature model, since the coauthorship relationship is symmetric, we learned a full, symmetric weight matrix $W$ as described in Section 2.4. We did not use any covariates. A visualization of the predictions for each of these algorithms can be seen in Figure 2(b-d). Figure 2 really drives home the difference in expressiveness. Stochastic blockmodels are required to group authors into classes, and assumes that all members of classes interact similarly. For visualization, we have ordered the authors by the groups the IRM found. These groups can clearly be seen in Figure 2(c). The MMSB, by allowing partial membership is not as restrictive. However, on this dataset, the IRM outperformed it. The latent feature relational model is the most expressive of the models and is able to much more faithfully reproduce the coauthorship network.

The latent feature relational model also quantitatively outperformed the IRM and MMSB. We again ran our sampler for 1000 samples initializing with either a random feature matrix or a class-based feature matrix from the IRM and reported the AUC on the held-out data. Using five restarts for each method, the LFRM w/ IRM performed best with an AUC of 0.9509, the LFRM rand was next with 0.9466 and much lower were the IRM at 0.8906 and the MMSB at 0.8705 (all at most $\pm 0.013$). On average, the latent feature relational model inferred 20-22 features when initialized with the IRM and 38-44 features when initialized randomly.

## 5  Conclusion

We have introduced the nonparametric latent feature relational model, an expressive nonparametric model for inferring latent binary features in relational entities. This model combines approaches from the statistical network analysis community, which have emphasized feature-based methods for analyzing network data, with ideas from Bayesian nonparametrics in order to simultaneously infer the number of latent binary features at the same time we infer the features of each entity and how those features interact. Existing class-based approaches infer latent structure that is a special case of what can be inferred by this model. As a consequence, our model is strictly more expressive than these approaches, and can use the solutions produced by these approaches for initialization. We showed empirically that the nonparametric latent feature model performs well at link prediction on several different datasets, including datasets that were originally used to argue for class-based approaches. The success of this model can be traced to its richer representations, which make it able to capture subtle patterns of interaction much better than class-based models.

**Acknowledgments**  KTM was supported by the U.S. Department of Energy contract DE-AC52-07NA27344 through Lawrence Livermore National Laboratory. TLG was supported by grant number FA9550-07-1-0351 from the Air Force Office of Scientific Research.

# References

[1] Stanley Wasserman and Katherine Faust. *Social Network Analysis: Methods and Applications*. Cambridge University Press, 1994.

[2] Stanley Wasserman and Philippa Pattison. Logit models and logistic regressions for social networks: I. an introduction to Markov random graphs and p$^*$. *Psychometrika*, 61(3):401–425, 1996.

[3] Garry Robins, Tom Snijders, Peng Wang, Mark Handcock, and Philippa Pattison. Recent developments in exponential random graph (p*) models for social networks. *Social Networks*, 29(2):192–215, May 2007.

[4] Yuchung J. Wang and George Y. Wong. Stochastic blockmodels for directed graphs. *Journal of the American Statistical Association*, 82(397):8–19, 1987.

[5] Krzysztof Nowicki and Tom A. B. Snijders. Estimation and prediction for stochastic blockstructures. *Journal of the American Statistical Association*, 96(455):1077–1087, 2001.

[6] Charles Kemp, Joshua B. Tenenbaum, Thomas L. Griffiths, Takeshi Yamada, and Naonori Ueda. Learning systems of concepts with an infinite relational model. In *Proceedings of the American Association for Artificial Intelligence (AAAI)*, 2006.

[7] Zhao Xu, Volker Tresp, Kai Yu, and Hans-Peter Kriegel. Infinite hidden relational models. In *Proceedings of the 22nd Conference on Uncertainty in Artificial Intelligence (UAI)*, 2006.

[8] Edoardo M. Airoldi, David M. Blei, Eric P. Xing, and Stephen E. Fienberg. Mixed membership stochastic block models. In D. Koller, Y. Bengio, D. Schuurmans, and L. Bottou, editors, *Advances in Neural Information Processing Systems (NIPS) 21*. Red Hook, NY: Curran Associates, 2009.

[9] Peter D. Hoff, Adrian E. Raftery, and Mark S. Handcock. Latent space approaches to social network analysis. *Journal of the American Statistical Association*, 97(460):1090–1098, 2002.

[10] Peter D. Hoff. Bilinear mixed-effects models for dyadic data. *Journal of the American Statistical Association*, 100(469):286–295, 2005.

[11] Peter D. Hoff. Multiplicative latent factor models for description and prediction of social networks. *Computational and Mathematical Organization Theory*, 2008.

[12] Thomas L. Griffiths and Zoubin Ghahramani. Infinite latent feature models and the Indian Buffet Process. In Y. Weiss, B. Schölkopf, and J. Platt, editors, *Advances in Neural Information Processing Systems (NIPS) 18*. Cambridge, MA: MIT Press, 2006.

[13] Edward Meeds, Zoubin Ghahramani, Radford Neal, and Sam Roweis. Modeling dyadic data with binary latent factors. In B. Schölkopf, J. Platt, and T. Hofmann, editors, *Advances in Neural Information Processing Systems (NIPS) 19*. Cambridge, MA: MIT Press, 2007.

[14] Daniel L. Navarro and Thomas L. Griffiths. Latent features in similarity judgment: A nonparametric Bayesian approach. *Neural Computation*, 20(11):2597–2628, 2008.

[15] Christian P. Robert and George Casella. *Monte Carlo Statistical Methods*. Springer, 2004.

[16] Radford M. Neal. Markov chain sampling methods for Dirichlet process mixture models. *Journal of Computational and Graphical Statistics*, 9(2):249–265, 2000.

[17] James H. Albert and Siddhartha Chib. Bayesian analysis of binary and polychotomous response data. *Journal of the American Statistical Association*, 88(422):669–679, 1993.

[18] Dilan Görür, Frank Jäkel, and Carl Edward Rasmussen. A choice model with infinitely many latent features. In *Proceedings of the 23rd International Conference on Machine learning (ICML)*, 2006.

[19] Rudolph J. Rummel. Dimensionality of nations project: Attributes of nations and behavior of nation dyads, 1950–1965. ICPSR data file, 1999.

[20] Woodrow W. Denham. *The Detection of Patterns in Alyawarra Nonverbal Behavior*. PhD thesis, University of Washington, 1973.

[21] Jin Huang and Charles X. Ling. Using AUC and accuracy in evaluating learning algorithms. *IEEE Transactions on Knowledge and Data Engineering*, 17(3):299–310, 2005.

[22] Amir Globerson, Gal Chechik, Fernando Pereira, and Naftali Tishby. Euclidean embedding of co-occurrence data. *The Journal of Machine Learning Research*, 8:2265–2295, 2007.

